# USE OF MULTI-LAYERED NETWORKS FOR CODING SPEECH WITH PHONETIC FEATURES

Yoshua Bengio, Regis Cardin
and Renato De Mori
Computer Science Dept.
McGill University
Montreal, Canada H3A2A7

Piero Cosi
Centro di Studio per le
Ricerche di Fonetica, C.N.R.,
Via Oberdan,10,
35122 Padova, Italy

## ABSTRACT

Preliminary results on speaker-independant speech recognition are reported. A method that combines expertise on neural networks with expertise on speech recognition is used to build the recognition systems. For transient sounds, event-driven property extractors with variable resolution in the time and frequency domains are used. For sonorant speech, a model of the human auditory system is preferred to FFT as a front-end module.

## INTRODUCTION

Combining a structural or knowledge-based approach for describing speech units with neural networks capable of automatically learning relations between acoustic properties and speech units is the research effort we are attempting. The objective is that of using good generalization models for learning speech units that could be reliably used for many recognition tasks without having to train the system when a new speaker comes in or a new task is considered.

Domain (speech recognition) specific knowledge is applied for
- **segmentation and labeling** of speech,
- definition of **event-driven property extractors**,
- use of an **ear model** as preprocessing applied to some modules,
- coding of network outputs with phonetic features,
- **modularization** of the speech recognition task by dividing the workload into smaller networks performing simpler tasks.
Optimization of **learning time** and of **generalization** for the neural networks is sought through the use of neural networks techniques :
- use of error back-propagation for learning,

- switching between on-line learning and batch learning when appropriate,
- convergence acceleration with **local** (weight specific) **learning rates**,
- convergence acceleration with **adaptive learning rates** based on information on the changes in the direction of the gradient,
- control of the presentation of examplars in order to **balance examplars** among the different classes,
- training of **small modules** in the first place:
    - simpler architecture (e.g. first find out the solution to the linearly separable part of the problem),
    - use of simple recognition task,

combined using either Waibel's *glue units* [Waibel 88] or with simple heuristics.
- training on time-shifted inputs to learn **time invariance** and insensitivity to errors in the segmentation preprocessing.
- controlling and improving generalization by using several test sets and using one of them to decide when to stop training.

## EAR MODEL

In recent years basilar membrane, inner cell and nerve fiber behavior have been extensively studied by auditory physiologists and neurophysiologists and knowledge about the human auditory pathway has become more accurate [Sachs79,80,83][Delgutte 80,84][Sinex 83]. The computational scheme proposed in this paper for modelling the human auditory system is derived from the one proposed by **S. Seneff** [Seneff 84,85,86]. The overall system structure which is illustrated in Fig. 1 includes three blocks: the first two of them deal with peripheral transformations occurring in the early stages of the hearing process while the third one attempts to extract information relevant to perception. The first two blocks represent the **periphery** of the earing system. They are designed using knowledge of the rather well known responses of the corresponding human auditory stages [Delgutte 84]. The third unit attempts to apply a useful processing strategy for the extraction of important speech properties like spectral lines related to **formants**.

The speech signal, band-limited and sampled at 16 kHz, is first pre-filtered through a set of four complex zero pairs to eliminate the very high and very low frequency components. The signal is then analyzed by the first block, a 40-channel **critical-band linear filter bank**. Filters were designed to optimally fit physiological data [Delgutte 84] such as those observed by [N.Y.S. Kiang et al.] and are implemented as a

cascade of complex high frequency zero pairs with taps after each zero pair to individual tuned resonators. The second block of the model is called the **hair cell synapse model**, it is nonlinear and is intended to capture prominent features of the transformation from basilar membrane vibration, represented by the outputs of the filter bank, to probabilistic response properties of auditory nerve fibers. The outputs of this stage, in accordance with [Seneff 88], represent the **probability of firing** as a function of time for a set of similar fibers acting as a group. Four different neural mechanisms are modeled in this nonlinear stage. The rectifier is applied to the signal to simulate the high level distinct directional sensitivity present in the inner hair cell current response. The short-term adaptation which seems due to the neurotransmitter release in the synaptic region between the inner hair cell and its connected nerve fibers is simulated by the "membrane model". The third unit represents the observed gradual loss of synchrony in nerve fiber behaviour as stimulus frequency is increased. The last unit is called "Rapid Adaptation", it performs "Automatic Gain Control" and implements a model of the refractory phenomenon of nerve fibers. The third and last block of the ear model is the **synchrony detector** which implements the known "phase locking" property of the nerve fibers. It enhances spectral peaks due to vocal tract **resonances**.

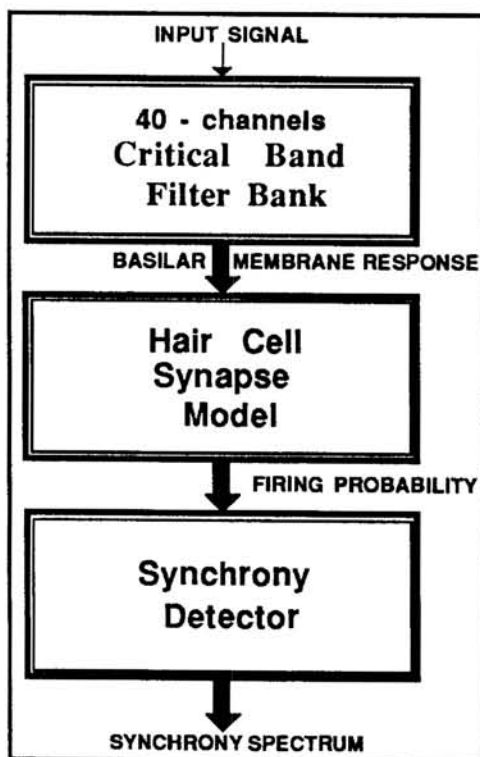

Figure 1 : Structure of the ear model

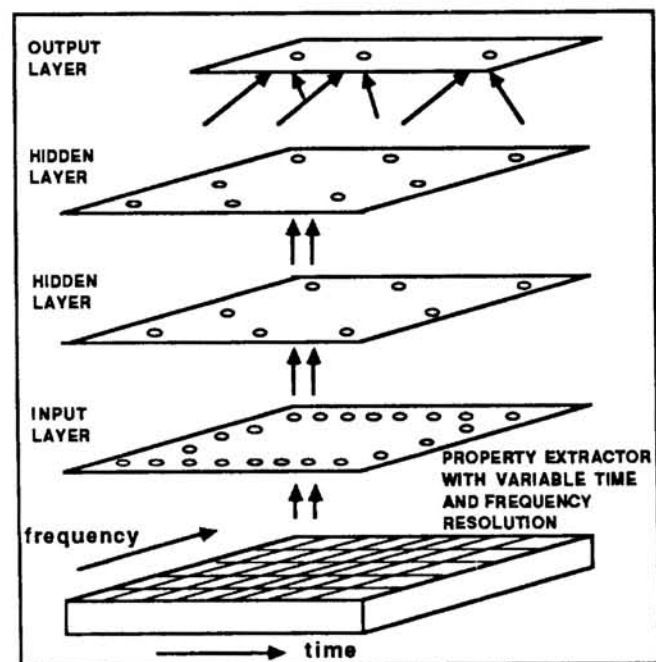

Figure 2 : Multi-layered network with variable resolution Property Extractor

## PROPERTY EXTRACTORS

For many of the experiments described in this paper, learning is performed by a **set** of multi-layered neural networks (MLNs) whose execution is decided by a **data-driven strategy.** This strategy analyzes morphologies of the input data and selects the execution of one or more MLNs as well as the **time and frequency resolution** of the spectral samples that are applied at the network input. An advantage of using such specialized property extractors is that the number of necessary input connections (and thus of connections) is then minimized, thus **improving the generalizing power** of the MLNs. Fine time resolution and gross frequency resolution are used, for example, at the onset of a peak of signal energy, while the opposite is used in the middle of a segment containing broad-band noise. The latter situation will allow the duration of the segment analyzed by one instantiation of the selected MLN to be larger than the duration of the signal analyzed in the former case.

Property extractors (PEs) are mostly rectangular windows subdivided into cells, as illustrated in Figure 2. Property extractors used in the experiments reported here are described in [Bengio, De Mori & Cardin 88]. A set of PEs form the input of a network called MLN1, executed when a situation characterized by the following rule is detected:

SITUATION $S_1$

        ((deep_dip)(t*)(peak))
  or   ((ns)(t*)(peak))
  or (deep_dip)(sonorant-head)(t*)(peak))
                          --> execute (MLN1 at t*)

(deep_dip), (peak), (ns) are symbols of the PAC alphabet representing respectively a deep dip, a peak in the time evolution of the signal energy and a segment with broad-band noise; t* is the time at which the first description ends, sonorant-head is a property defined in [De Mori, Merlo et al. 87]. Similar preconditions and networks are established for nonsonorant segments at the end of an utterance.

Another MLN called MLN2 is executed only when frication noise is detected. This situation is characterized by the following rule:

SITUATION $S_2$

(pr1= (ns)) --> execute (MLN2 every  T=20 msecs.)

# EXPERIMENTAL RESULTS

## EXPERIMENT 1

- task : perform the classification among the following 10 letters of the alphabet, from the **E-set** : { b,c,d,e,g,k,p,t,v,3}
- Input coding defined in [Bengio, De Mori & Cardin 88].
- architecture : two modules have been defined, MLN1 and MLN2. The input units of each PE window are connected to a group of 20 hidden units, which are connected to another group of 10 hidden units. All the units in the last hidden layer are then connected to the 10 output units.
- database : in the learning phase, 1400 samples corresponding to 2 pronounciations of each word of the E-set by 70 speakers were used for training MLN1 and MLN2. Ten **new speakers** were used for testing. The data base contains 40 male and 40 female speakers.
- results :  an overall error rate of 9.5% was obtained with a maximum error of 20% for the letter /d/. These results are much better than the ones we obtained before and we published recently [De Mori, Lam & Gilloux 87]. An observation of the confusion matrix shows that most of the errors represent cases that appear to be difficult even in human perception.

## EXPERIMENT 2

- task : similar to the one in experiment 1 i.e. to recognize the **head consonant** in the context of a certain vowel : /ae/,/o/,/u/ and /a/.
-subtask 1 : classify pronounciations of the first phoneme of letters A,K,J,Z and digit 7 into the classes {/vowel/,/k/,/j/,/z/,/s/}.
-subtask 2 : classify pronounciations of the first phoneme of letter O and digit 4 into the classes {/vowel/,/f/}.
-subtask 3 : classify pronounciations of the first phoneme of the letter Y and the digits 1 and 2 into the classes {/vowel/,/t/}.
-subtask 4 : classify pronounciations of the first phoneme of letters I,R,W and digits 5 and 9 into the classes {/vowel/,/d/,/f/,/n/}
- input coding : as for experiment 1 except that only PEs pertaining to situation S1 were used, as the input to a single MLN.
- architecture : two layers of respectively 40 and 20 hidden units followed by an output unit for each of the classes defined for the subtask.
- database : 80 speakers (40 males, 40 females) each pronouncing two utterances of each letter and each digit. The first 70 speakers are used for training, the last 10 for testing.
- results :
subtask 1 : {/vowel/,/k/,/j/,/z/,/s/} preceding vowel /ae/.
        4 % error on test set.

subtask 2 : {/vowel/,/f/} preceding vowel /o/.
        0 % error on test set.
subtask 3 : {/vowel/,/t/} preceding vowel /u/.
        0 % error on test set.
subtask 4 : {/vowel/,/d/,/f/,/n/} preceding vowel /a/.
        3 % error on test set.

# EXPERIMENT 3

- task : **speaker-independant vowel recognition** to discrimine among ten vowels extracted from 10 english words : {BEEP,PIT,BED,BAT,BUT,FUR,FAR,SAW,PUT,BOOT}.
- input coding : the signal processing method used for this experiment is the one described in the section "ear model". The output of the Generalised Synchrony Detector (GSD) was collected every 5 msecs. and represented by a 40-coefficients vector. Vowels were automatically singled out by an algorithm proposed in [De Mori 85] and a linear interpolation procedure was used to reduce to 10 the variable number of frames per vowel (the first and the last 20 ms were not considered in the interpolation procedure).
- architecture : 400 input units (10 frames x 40 filters), a single hidden layer with 20 nodes, 10 output nodes for the ten vowels.
- database : speech material consisted in 5 pronounciations of the ten monosyllabic words by 13 speakers (7 male, 6 female) for training and 7 new speakers (3 male, 4 female) for test.
- results : In **95.4%** of the cases, correct hypotheses were generated with the highest evidence, in 98.5% of the cases correct hypotheses were found in the top two candidates and in 99.4 % of the cases in the top three candidates. The same experiment with FFT spectra instead of data from the ear model gave 87% recognition rate in similar experimental conditions. The use of the ear model allowed to produce spectra with a limited number of well defined spectral lines. This represents a good use of speech knowledge according to which formants are vowel parameters with low variance. The use of male and female voices allowed the network to perform an excellent generalization with samples from a limited number of speakers.

# CONCLUSION

The preliminary experiments reported here on speaker normalization combining multi-layered neural networks and speech recognition expertise show promising results. For transient sounds, event-driven property extractors with variable resolutions in the time and frequency domains were used. For sonorant speech with formants, a new model of

the human auditory system was preferred to the classical FFT or LPC representation as a front-end module. More experiments have to be carried out to build an integrated speaker-independant phoneme recognizer based on multiple modules and multiple front-end coding strategies. In order to tune this system, variable depth analysis will be used. New small modules will be designed to specifically correct the deficiencies of trained modules. In addition, we consider strategies to perform recognition at the word level, using as input the sequence of outputs of the MLNs as time flows and new events are encountered. These strategies are also useful to handle slowly varying transitions such as those in diphtongs.

## REFERENCES

Bengio Y., De Mori R. & Cardin R., (1988)"Data-Driven Execution of Multi-Layered Networks for Automatic Speech Recognition", Proceedings of AAAI 88, August 88, Saint Paul, Minnesota,pp.734-738.

Bengio Y. & De Mori R. (1988), "Speaker normalization and automatic speech recognition using spectral lines and neural networks", Proceedings of the Canadian Conference on Artificial Intelligence (CSCSI-88), Edmonton, Al., May 1988.

Delgutte B. (1980), "Representation of speech-like sounds in the discharge patterns of auditory-nerve fibers" , Journal of the Acoustical Society of America, N. 68, pp. 843-857.

Delgutte B. & Kiang N.Y.S. (1984) , "Speech coding in the auditory nerve", Journal of Acoustical Society of America , N. 75, pp. 866-907.

De Mori R., Laface P . & Mong Y. (1985), "Parallel algorithms for syllable recognition in continuous speech", IEEE Transactions on Pattern Analysis and Machine Intelligence, Vol. PAMI-7, N. 1, pp. 56-69, 1985.

De Mori R., Merlo E., Palakal M. & Rouat J.(1987),"Use of procedural knowledge for automatic speech recognition", Proceedings of the tenth International Joint Conference on Artificial Intelligence, Milan, August 1987, pp. 840-844.

De Mori R., Lam L. & Gilloux M. (1987), "Learning and plan refinement in a knowledge-based system for automatic speech recognition", IEEE Transactions on Pattern Analysis and Machine Intelligence, vol. PAMI-9, No.2, pp.289-305.

Kiang N.Y.S., Watanabe T., Thomas E.C. & Clark L.F., "Discharge patterns of single fibers in the cat's auditory-nerve fibers", Cambridge, MA: MIT press.

Rumelhart D.E., Hinton G.E. & Williams R.J. (1986),"Learning internal representation by error propagation", Parallel Distributed Processing : Exploration in the Microstructure of Cognition, vol. 1, pp.318-362, MIT Press, 1986.

Seneff S. (1984), "Pitch and spectral estimation of speech based on an auditory synchrony model", Proceedings of ICASSP-84, San Diego, CA.

Seneff S. (1985), "Pitch and spectral analysis of speech based on an auditory synchrony model", RLE Technical Report 504 , MIT.

Seneff S. (1986), "A computational model for the peripheral auditory system: application to speech recognition research", Proceedings of ICASSP-86, Tokyo, pp. 37.8.1-37.8.4.

Seneff S. (1988), "A joint synchrony/mean-rate model of auditory speech processing", Journal of Phonetics, January 1988.

Sachs M.B. & Young E.D. (1979),"Representation of steady-state vowels in the temporal aspects of the discharge pattern of populations of auditory nerve fibers", Journal of Acoustical Society of America, N. 66, pp. 1381-1403.

Sachs M.B. & Young E.D. (1980),"Effects of nonlinearities on speech encoding in the auditory nerve", Journal of Acoustical Society of America, N. 68, pp. 858-875.

Sachs M.B. & Miller M.I. (1983), "Representation of stop consonants in the discharge patterns of auditory-nerve fibers", Journal of Acoustical Society of America, N. 74, pp. 502-517.

Sinex D.G. & Geisler C.D. (1983), "Responses of auditory-nerve fibers to consonant-vowel syllables", Journal of Acoustical Society of America, N. 73, pp. 602-615.

Waibel A. (1988),"Modularity in Neural Networks for Speech Recognition", Proc. of the 1988 IEEE Conference on Neural Information Processing Systems, Denver, CO.